# Nonlinear physically-based models for decoding motor-cortical population activity

**Gregory Shakhnarovich**  **Sung-Phil Kim**  **Michael J. Black**
Department of Computer Science
Brown University
Providence, RI 02912
{gregory,spkim,black}@cs.brown.edu

## Abstract

Neural motor prostheses (NMPs) require the accurate decoding of motor cortical population activity for the control of an *artificial motor system*. Previous work on cortical decoding for NMPs has focused on the recovery of hand kinematics. Human NMPs however may require the control of computer cursors or robotic devices with very different physical and dynamical properties. Here we show that the firing rates of cells in the primary motor cortex of non-human primates can be used to control the parameters of an artificial physical system exhibiting realistic dynamics. The model represents 2D hand motion in terms of a point mass connected to a system of idealized springs. The nonlinear spring coefficients are estimated from the firing rates of neurons in the motor cortex. We evaluate linear and a nonlinear decoding algorithms using neural recordings from two monkeys performing two different tasks. We found that the decoded spring coefficients produced accurate hand trajectories compared with state-of-the-art methods for direct decoding of hand kinematics. Furthermore, using a physically-based system produced decoded movements that were more "natural" in that their frequency spectrum more closely matched that of natural hand movements.

## 1 Introduction

Neural motor prostheses (NMPs) aim to restore lost motor function to people with intact cerebral motor areas who, through disease or injury, have lost the ability to control their limbs. Central to these devices is a method for *decoding* the firing activity of motor cortical neurons to produce a voluntary control signal. A number of groups have recently demonstrated the real-time neural control of 2D or 3D computer cursors or simple robotic limbs in monkeys [1, 13, 18, 20, 22] and humans [6]. Previous work on decoding motor cortical signals however has focused on modeling the relationship between neural firing rates and simple hand kinematics including hand direction, speed, position, velocity, or acceleration [2, 4, 8, 10].

While the relationship between neural firing rates and hand kinematics is well established in able-bodied monkeys, the situation of a human NMP is quite different. For a paralyzed human, the NMP represents an *artificial motor system* with different physical properties than the intact human motor system. In particular, a human NMP may involve the control of devices as different as computer cursors or robotic wheelchairs. It remains an open question whether motor cortical neurons can successfully control such varied systems with dynamics that are quite different from human limbs.

Here we propose a model that makes a first step toward neural control of novel artificial motor systems. We show that motor cortical firing rates can be nonlinearly related to the parameters of an idealized physical system. This provides an important proof-of-concept for human NMPs. Our model decodes the *dynamics* of hand movement directly from the neural activity. Ultimately, such a

model should reflect the actuator being controlled. For a biological actuator this means the activation of individual muscles; for a robotic one, the forces and torques produced by the motors in the system.

A model incorporating direct cortical control of dynamics has been proposed in [19]. There are two major distinctions between that work and ours. First, we consider the task of controlling an artificial system, rather than the subject's real limb. Second, applying the model in [19] in practice would require constructing a very complex biomechanical model and controlling its many degrees of freedom with a limited neural bandwidth. Here we propose a much simpler approach, that does not attempt to accurately model the musculoskeletal structure of the arm. Instead, it provides a computationally effective framework to model the dynamics of the limb moving in two dimensional plane. Our approach is inspired by the recent work of Hinton and Nair [5], that suggested a generative model for images of handwritten digits. In that work, observed images were assumed to have been generated by a pen connected to a set of springs, the trajectory of the pen controlled by varying the stiffness of the springs according to a digit-specific "motor program". The goal was to infer the motor program from an observed image, in order to classify the digit. In the context of neural decoding, the image observation is replaced with the recorded neural signal, from which we need to recover the "motor program", and thus the intended movement. This is where the parallels between our work and [5] end. One particularly important difference is that the neural decoder may be learned in a supervised procedure, where the groud truth for the movement associated with a given neural signal is known.

An advantage of this spring-based model (SBM) over previous kinematics-based decoding methods is that the realistic dynamics of the model produce smoother recovered movement. We show that the motions are more natural in that they better match the power spectrum of true hand movements. This suggests that the control of a physical system (even an artificial one) may prove more natural for a human NMP.

The experimental setup we consider in this paper involves an electrode array, implanted in the arm/hand area of the MI cortex of a behaving monkey [17]. The animals are trained to control the cursor by moving the endpoint of a two-link manipulandum constrained to a plane, much like a human would use a computer mouse [11, 13]. Neural data and hand kinematics were recorded from two monkeys performing two different tasks. The data was separated into training and testing segments and we quantitatively compared a variety of popular linear and nonlinear algorithms for decoding hand kinematics and the spring coefficients of our SBM. As expected, nonlinear methods tend to outperform linear ones. Moreover, movement reconstructed with the SBM has a power spectrum significantly closer to that of natural movement. These results suggest that the control of idealized physical systems with real-time nonlinear decoding algorithms may form the basis for a practical human NMP.

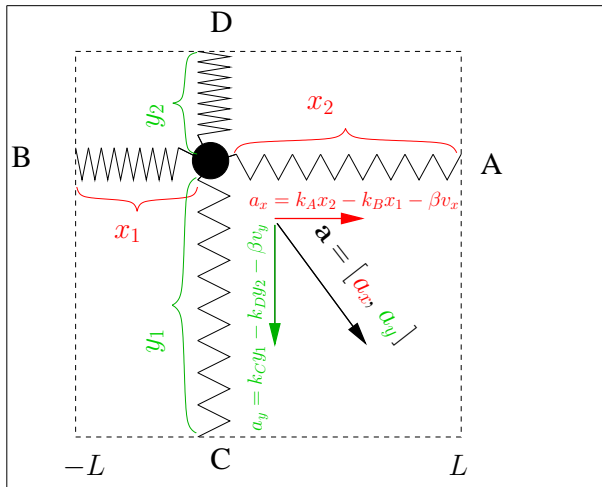

Figure 1: Sketch of the spring-based model. The outer endpoints of the springs are assumed to slide without friction, so that A and B are always orthogonal to C and D. The rest length is assumed to be zero for all springs. Movement is controlled by varying the stiffness coefficients $k_A, k_B, k_C$ and $k_D$.

## 2 The spring-based model

Decoding neural activity in $N$ cells involves estimating the values of a hidden state $\mathbf{X}(t)$ at time $t$ given an observed sequence of firing rates $\mathbf{Z}(0)\ldots\mathbf{Z}(t)$ up to time $t$, with each $\mathbf{Z}(i)$ being a $1\times N$ vector. The state here is typically taken to be either hand position, velocity, etc. Methods described in the literature can be roughly divided into two classes. *Generative* methods formulate the likelihood of the observed firing rates conditioned on the state and use Bayesian inference methods such as the Kalman filter [21] or particle filter [3] to estimate the system state from observations. In contrast, direct (or *discriminative*) methods learn a function that maps firing rates over some preceding temporal window into hand kinematics. Various methods have been explored including linear regression [1, 13], support-vector regression [15] and neural network algorithms [12, 20]. All these previous methods have focused on direct decoding of kinematic properties of the hand movement and have ignored the arm *dynamics*.

### 2.1 Parametrization of dynamics

Our approach to incorporating dynamics into the decoding process has been inspired by the following model of [5], sketched out in Figure 1. Without loss of generality, let the work area (that fully contains the movement range) be an axis-aligned square $[-L, L] \times [-L, L]$. The endpoint of the limb (wrist) is assumed to be connected to one end of four imaginary springs, the other end of which is sliding with no friction along rails forming the boundaries of the "work area". Thus, at every time instance each spring is parallel to one of the axes. The analysis of dynamics therefore can be easily decomposed to $x$ and $y$ components. Below we focus on the $x$ component.

All four springs are assumed to have rest length of zero. Suppose that the position of the wrist at time $t$ is $[x(t), y(t)]$. Then the springs $A$ and $B$ apply forces determined by Hooke's law, namely, $k_A(t)\,(L - x(t)))$ and $-k_B(t)\,(x(t) + L)$, where $k_A(t)$ and $k_B(t)$ are the stiffness coefficients of $A$ and $B$ at time $t$. To reflect physical constraints on movement in the real world, the model presumes a point mass $m$ in the center of the wrist (i.e. at the cursor location). Furthermore, it is assumed that the movement is damped by a viscous force proportional to the instantaneous velocity, $-\beta v_x(t)$. The viscosity is meant to represent both the medium resistance and the elasticity of the muscles. In summary, according to Newton's second law the acceleration of hand at time $t$ is given by

$$m \cdot a_x(t) \;=\; k_A(t) \cdot (L - x(t)) \;-\; k_B(t) \cdot (L + x(t)) - \beta \cdot v_x(t), \tag{1}$$

where $v_x(t)$ is the instantaneous velocity of the wrist at time $t$ along the $x$ axis.

Control of movement in this model is realized through varying the stiffness coefficients of the springs: given the current position of the wrist $x$, the desired acceleration $a$ is achieved by setting $k_A(t)$ and $k_B(t)$ so as to solve (1). This solution is not unique, in general. We note, however, that the physiological meaning of the $k$'s requires them to be non-negative, since the muscles can not "push". This motivates us to introduce the total stiffness constraint

$$k_A + k_B = \kappa, \tag{2}$$

where $\kappa$ is a constant chosen so that no feasible acceleration would yield negative $k_A$ or $k_B$.

We can now recover the underlying parameters $\mathbf{K} = [k_A, k_B, k_C, k_D]$ for the observed movement by applying (1) at each time step as follows. First we estimate the velocities $\hat{v}_x(t) = x(t+1) - x(t)$ and accelerations $\hat{a}_x(t) = \hat{v}_x(t+1) - \hat{v}_x(t)$. Then, we substitute (2) into (1), yielding

$$\hat{k}_A(t) = \frac{m \cdot \hat{a}_x(t) + \hat{v}_x(t) + \kappa \cdot (L + x(t))}{2L}. \tag{3}$$

The value of $k_B(t)$ is then uniquely determined from (2). Repeating these calculations for the $y$-axis produces the coefficients for springs $C$ and $D$.

### 2.2 Decoding neural activity

We now turn to our main goal: inferring the desired movement from a recorded neural signal. We treat this as a supervised learning task. In the training stage, we take a data set in which we have both the recorded neural signal $\mathbf{Z}(t)$ and the observed trajectory of hand positions $\mathbf{X}(t)$ associated

with that signal. From this, we can learn a *mapping* $g$ from the neural signal to the desired representation of movement. For direct kinematic decoding this means inference $g : \mathbf{Z}(t) \rightarrow \mathbf{X}(t)$. For decoding with the SBM, this means inference of spring coefficients in the SBM, $g : \mathbf{Z}(t) \rightarrow \mathbf{K}(t)$, followed by the calculation $\mathbf{K}(t) \rightarrow \mathbf{X}(t)$ as described above. The SBM formulation also requires a preprocessing step for the training data: we need to convert the observed position trajectory $\mathbf{X}$ to the trajectory through $\mathbf{K}$, acording to (3).

We have focused on two ways of constructing $g$, described below.

**Linear filter.** The linear filter (LF) approach [13] consists of modeling the mapping from firing rate to movement by a linear transformation $\mathbf{W}$ that is applied on a concatenated firing rate vector for a fixed *history depth* $l$:

$$\mathbf{X}(t) = \mathbf{x}_0 + \mathbf{W}\tilde{\mathbf{Z}}(t), \tag{4}$$

where $\mathbf{x}_0$ is a constant (bias) term and

$$\tilde{\mathbf{Z}}(t) = \left[ \mathbf{Z}^T(t-l), \ldots, \mathbf{Z}^T(t) \right]^T. \tag{5}$$

The dimension of $\tilde{\mathbf{Z}}(t)$ for a recording from $N$ channels, is $1 \times lN$. The transformation $\mathbf{W}$ is fit to the training data by solving the least squares problem, and then used at the decoding stage to predict values of $\mathbf{X}$. Application of the LF to the SBM is straightforward: the target of the mapping is in the space of coefficients $\mathbf{K}$, rather than position $\mathbf{X}$.

**Support vector regression.** Support Vector Machines (SVM) are a widely popular learning architecture that relies on two key ideas: mapping the data into a (possibly infinite-dimensional) feature space using a *kernel* function, and optimizing the bound on generalization error. In the context of regression [16] this means using an $\epsilon$-insensitive loss function, that does not penalize training errors up to $\epsilon$, to fit a linear function in the feature space. SVMs also aim at reducing model complexity by penalizing the objective for the norm of the resulting function. The solution is finally expressed in terms of kernel functions involving a subset of the training examples (the *support vectors*). The key parameters that affect the performance of SVMs are the value of $\epsilon$, the tradeoff $c$ that governs the penalty of training error, and parameters of the kernel function.

SVMs have been widely successful in many applications of machine learning. However, their application to the task of neural decoding has been limited to the directional center-out task [15]. Here we evaluate SV regression as a method for decoding more general 2D movement. Again, the SVM formulation is readily extended to the SBM (with the target functions being components of $\mathbf{K}$).

**Alternative decoders.** A variety of other decoding approaches has been proposed in the literature. We conducted experiments with three additional algorithms: Kalman filter [21], Multilayer Perceptrons [20] and Echo-state Networks, a recurrent neural network architecture [7]. The Kalman filter uses a linear model of the mapping of neural signals to movement, while the models underlying the other two methods are nonlinear. Our findings can be summarized as follows, for both kinematic decoding and decoding with the spring-based model. First, nonlinear methods perform significantly better than linear ones. Second, there was a trend for the Kalman filter to perform better than the linear filter. Third, among nonlinear methods SVM tended to perform better than the two neural network architectures. However, these latter differences could not be established with significance. In the following section, we focus on experiments with the linear filter (the de-facto standard decoding method today) and SVM, which achieved the overall best results in our experiments.

## 3 Experiments

We evaluated the performance of the proposed approach on data sets obtained from two behaving monkeys (*Macaca Mulatta*). The neural signal was obtained with a Cyberkinetics microelectrode array [9] (96 electrodes) implanted in the arm/hand area of MI cortex. The experimental animals performed the tasks described below.

***Sequential* reaching movement**, described in [13] Reach targets and a hand position feedback cursor were presented on a video screen in front of the monkey. When a reach target was presented

| Session | Units | Train | Test |
|---|---|---|---|
| CL-sequential | 49 | 623 | 140 |
| LA-continuous | 96 | 244 | 165 |
| CL-continuous | 55 | 448 | 140 |

Table 1: Details of experiments. Units: number of distinct units identified after spike sorting. Train, test: length of train and test sequences in seconds.

the animal's task was to move a manipulandum so that the feedback cursor moved into the target and remained in the target for 500ms, at which time that target was extinguished and a new reach target was presented in a different location. Target locations were drawn i.i.d. from the uniform distribution over the screen surface. This was repeated for up to 10 targets per trial. Upon successful completion of a trial the animal received a juice reward. Hand kinematics and neural activity were simultaneously recorded while the animal performed the task.

***Continuous* tracking** , described in [14] Monkey was viewing a computer screen on which a visual target appeared in a random, but smooth, sequence of locations. The monkey was trained to follow the target's position with a cursor, using a manipulandum, and received a reward for each successful trial (i.e. when the cursor remained within the target for a duration drawn for each target randomly between 3 and 10 seconds).

The recorded neural activity was converted to spike trains by computer-assisted spike-sorting software, and the spike counts were calculated in non-overlapping 70ms windows. The hand kinematics (obtained by recording the 2D position of the manipulandum) were averaged within each window, to produce an aligned representation.

## 3.1 Evaluation protocol

In each of the data sets, we selected a segment of the recording to train all the decoders, and a subsequent segment to test the decoding accuracy. Tuning of parameters (the kernel parameters of the SVM or the mass and viscosity of the spring model) was done on a held-out portion of the training segment. We built the firing rate history matrix by concatenating for each time step the firing rates for 15 bins. For instance, for monkey CL, continuous tracking, the dimension of the neural signal representation was 825 (55 channels $\times$ 15 history bins). This firing rates were then normalized so that all values would be within [-1,1]. Basic statistics of the data used in the experiments are given in Table 1.

We considered three evaluation criteria:

**Correlation coefficients** (CC): between the estimated and true value for each of the two spatial coordinates over the entire trajectory:

$$\text{CC} = \frac{\sum_t (x_t - \bar{x}_t)(\hat{x}_t - \bar{\hat{x}}_t)}{\sqrt{\sum_t (x_t - \bar{x}_t)^2 \sum_t (\hat{x}_t - \bar{\hat{x}}_t)^2}}.$$

**Mean absolute error** (MAE): in the estimated position versus the ground truth: $\text{MAE} = \frac{1}{N}\sum_{t=1}^{N} \|\mathbf{X}(t) - \hat{\mathbf{X}}(t)\|$.

**Power spectrum reconstruction** : One of the objectives of a practical decoding algorithm, especially in the context of assistive technology, is to produce movement that appears "natural". As a criterion for evaluating the degree of "naturalness" we use the similarity between power spectrum densities of the true movement and the reconstructed one. Specifically, we calculated the $L_1$ norm between the energy distributions over normalized angular frequencies, taken in the log domain (see Figure 2 for illustration).

## 3.2 Results

The reported results for SVM were obtained with quadratic kernel, $k(\mathbf{x}, \mathbf{y}) = (\mathbf{x} \cdot \mathbf{y} + 1)^2$; the trade-off term $c$ was fixed to 100, and the insensitivity parameter $\epsilon$ was set to 5 for the spring coefficient and 2 for direct position decoding. The number of support vectors was between 20% and 65% of the training set size.

| Decoder | CL/sequential | | | LA/continuous | | | CL/continuous | | |
|---|---|---|---|---|---|---|---|---|---|
| | MAE | $CC_x$ | $CC_y$ | MAE | $CC_x$ | $CC_y$ | MAE | $CC_x$ | $CC_y$ |
| Linear-kinematics | 5.3 | 0.69 | 0.79 | 5.03 | 0.5 | 0.75 | 6.66 | 0.80 | 0.83 |
| Linear-SBM | 5.7 | 0.64 | 0.74 | 5.26 | 0.46 | 0.72 | 6.82 | 0.77 | 0.81 |
| SVM-kinematics | 4.45 | 0.80 | 0.85 | 4.44 | 0.60 | 0.82 | 3.82 | 0.86 | 0.86 |
| SVM-SBM | 4.91 | 0.76 | 0.81 | 4.69 | 0.55 | 0.80 | 4.05 | 0.83 | 0.84 |

Table 2: Summary of results on the three datasets. MAE is given in cm, over workspace of roughly 30×30 cm.

Table 2 summarizes the MAE and CC measured on the test segment for each method. One observation is that SVM tends to outperform the linear filter, in line with previous observations [12, 15]. We believe that this is due to inherent nonlinearity in the underlying relationship, which is better captured by the SVM. Moreover, it is apparent that the decoding accuracy of the SBM is on par with that of the conventional kinematic decoding (the observed differences were not significant at the 0.05 level, measured over the per-bin position errors).

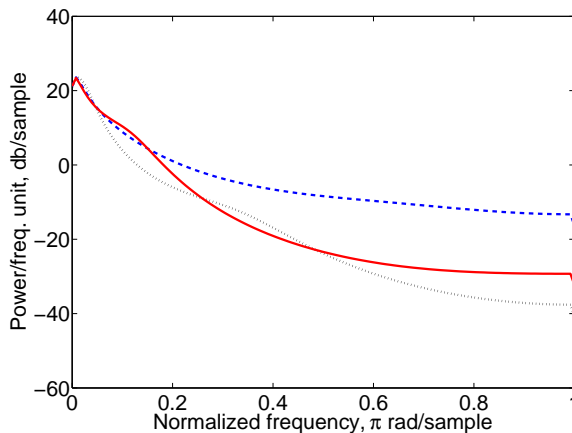

Figure 2: Example of power spectrum densities for true hand trajectory (dotted black), reconstruction with SVM-kinematics (dashed blue) and reconstruction with SVM-SBM (solid red). Estimated using Burg's algorithm (`pburg` in Matlab, order 4). Data from $x$ coordinate, LA-continuous.

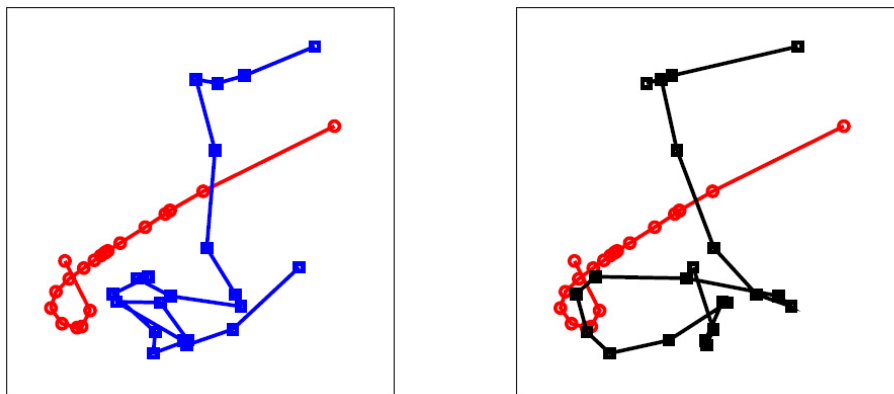

Figure 3: A 1.5 second path segment, true (circles) and reconstructed (squares). Left: SVM on kinematics, right: SVM with SBM. Markers show position averaged in each 70ms bin. Note the ragged form of the SVM-kinematics trajectory.

Results in Table 2, however, tell only a part of the story. Figure 3 shows, for a segment of 4.2 sec, a typical example of the movement reconstructed with SVM on kinematics versus SVM on spring coefficients. The accuracy in terms of deviation from ground truth is similar, however the estimate produced by the direct kinematic decoding is significantly more "ragged". Such discrepancy is not

necessarily reflected in the standard measures of accuracy such as CC or MAE. Quantitavely, this can be assessed by calculating the $L_1$ norm between the power spectrum densities of the true and reconstructed hand trajectories. The estmated values of this quantity in our experiments are shown in Table 3. These results reflect the relationship shown in Figure 2 (a typical case).

Table 3: Estimated $L_1$ norm between power spectrum density of true and reconstructed trajectories.

| Decoder | CL-sequential | | LA-continuous | | CL-continuous | |
|---|---|---|---|---|---|---|
| | $x$ | $y$ | $x$ | $y$ | $x$ | $y$ |
| Linear-kinematics | 147.41 | 154.80 | 199.24 | 206.61 | 49.68 | 44.37 |
| Linear-SBM | 71.58 | 68.24 | 72.99 | 80.37 | 35.68 | 43.72 |
| SVM-kinematics | 143.78 | 151.35 | 188.65 | 196.14 | 33.96 | 28.44 |
| SVM-SBM | 51.45 | 52.31 | 53.05 | 66.20 | 20.83 | 21.15 |

## 4 Discussion

The spring-based model proposed in this paper represents a first attempt to directly incorporate realistic physical constraints into a neural decoding model. Our experiments illustrate that the coefficients of an idealized physical system can be decoded from motor cortical firing rates, without sttistically significant loss of decoding accuracy compared to more standard direct decoding of kinematics. An advantage of such an approach is that the physical properties of the system damp high frequency motions resulting in decoded movements that inherently have the properties of natural movement, with no ad-hoc smoothing.

Future work should consider more sophisticated physical models such as a simulated robotic arm and a biophysically motivated musculoskeletal system. With the current state of the art in neural recording and decoding, recovering the parameters of such models may be challenging. In contrast, the approach presented here "summarizes" the effect of a more complicated system with just a few idealized muscle-like elements.

Additional experiments are also warranted. In particular using a robotic feedback device we can simulate the physical system of springs presented here such that the monkeys control a device with the properties of our model. We hypothesize that the accuracy of decoding spring coefficients from motor cortical activity in this condition will improve. This would suggest that matching the decoding model to the physical system being controlled will improve decoding accuracy.

Finally, the real test of physically-based models will come in human NMP experiments. We plan to test human cursor control with kinematic and physically-based decoders. We hypothesize that the dynamics of the physically-based model will make it easier to control accurately (and perhaps provide a more satisfying experience for the user). This could be a first step toward the neural control of mechanical actuators in the physical world.

## Acknowledgments

This work is partially supported by NIH-NINDS R01 NS 50867-01 as part of the NSF/NIH Collaborative Research in Computational Neuroscience Program and by the Office of Naval Research (award N0014-04-1-082). We also thank the European Neurobotics Program FP6-IST-001917. We thank Matthew Fellows and John Donoghue for providing data, and Reza Shadmehr for helpful conversations.

## References

[1] J. M. Carmena, M. A. Lebedev, R. E. Crist, J. E. O'Doherty, D. M. Santucci, D. F. Dimitrov, P. G. Patil, C. S. Henriquez, and M. A. L. Nicolelis. Learning to control a brain-machine interface for reaching and grasping by primates. *PLoS, Biology*, 1(2):001–016, 2003.

[2] D. Flament and J. Hore. Relations of motor cortex neural discharge to kinematics of passive and active elbow movements in the monkey. *Journal of Neurophysiology*, 60(4):1268–1284, 1988.

[3] Y. Gao, M. J. Black, E. Bienenstock, S. Shoham, and J. P. Donoghue. Probabilistic inference of hand motion from neural activity in motor cortex. In T. G. Dietterich, S. Becker, and Z. Ghahramani, editors, *Advances in Neural Information Processing Systems 14*, pages 213–220, Cambridge, MA, 2002. MIT Press.

[4] A. Georgopoulos, A. Schwartz, and R. Kettner. Neural population coding of movement direction. *Science*, 233:1416–1419, 1986.

[5] G. E. Hinton and V. Nair. Inferring motor programs from images of handwritten digits. In *Advances in Neural Information Processing*, 2005.

[6] L. R. Hochberg, J. A. Mukand, G. I. Polykoff, G. M. Friehs, and J. P. Donoghue. Braingate neuromotor prosthesis: Nature and use of neural control signals. In *Society for Neuroscience Abst. Program No. 520.17, Online*, 2005.

[7] H. Jaeger. The "echo state" approach to analyzing and training recurrent neural networks. Technical Report GMD Report 148, German National Research Institute for Computer Science, 2001.

[8] R. Kettner, A. Schwartz, and A. Georgopoulos. Primary motor cortex and free arm movements to visual targets in three-dimensional space. iii. positional gradients and population coding of movement direction from various movement origins. *Journal of Neuroscience*, 8(8):2938–2947, 1988.

[9] E. Maynard, C. Nordhausen, and R. Normann. The Utah intracortical electrode array: A recording structure for potential brain-computer interfaces. *Electroencephalography and Clinical Neurophysiology*, 102:228–239, 1997.

[10] D. Moran and A. Schwartz. Motor cortical representation of speed and direction during reaching. *Jrnl. of Neurophysiology*, 82(5):2676–2692, 1999.

[11] L. Paninski, M. Fellows, N. Hatsopoulos, and J. P. Donoghue. Spatiotemporal tuning of motor cortical neurons for hand position and velocity. *J. of Neurophysiology*, 91:515–532, 2004.

[12] Y. N. Rao, S.-P. Kim, J. Sanchez, D. Erdogmus, J. Principe, J. Carmena, M. Lebedev, and M. Nicolelis. Learning mappings in brain-machine interfaces with echo state networks. In *IEEE Int. Conf. on Acou., Speech, and Sig. Proc.*, volume 5, pages 233–236, March 2005.

[13] M. D. Serruya, N. G. Hatsopoulos, L. Paninski, M. R. Fellows, and J. P. Donoghue. Brain-machine interface: Instant neural control of a movement signal. *Nature*, 416:141–142, 2002.

[14] S. Shoham, L. M. Paninsky, M. R. Fellows, N. G. Hatsopoulos, J.P. Donoghue, and R. A. Normann. Statistical encoding model for a primary motor cortical brain-machine interface. *IEEE Transactions on Biomedical Engineering*, 52(7):1312–1322, 2005.

[15] L. Shpigelman, K. Crammerr, R. Paz, E. Vaadia, and Y. Singer. A temporal kernel-based model for tracking hand-movements from neural activities. In *Advances in Neural Information Processing*, Vancouver, BC, December 2005.

[16] A. J. Smola and B. Schölkopf. A tutorial on support vector regression. *Statistics and Computing*, 14:199–222, 2004.

[17] S. Suner, M. R. Fellows, C. Vargas-Irwin, G. K. Nakata, and J. P. Donoghue. Reliability of signals from a chronically implanted, silicon-based electrode array in non-human primate primary motor cortex. *IEEE Trans. on Neural Systems and Rehab. Eng.*, 13(4):524–541, 2005.

[18] D. Taylor, S. Helms Tillery, and A. Schwartz. Direct cortical control of 3D neuroprosthetic devices. *Science*, 296(5574):1829–1832, 2002.

[19] E. Todorov. Direct cortical control of muscle activation in voluntary arm movements: a model. *Nature Neuroscience*, 3(4):391–398, April 2000.

[20] J. Wessberg, C. Stambaugh, J. Kralik, Laubach M. Beck, P., J. Chapin, J. Kim, S. Biggs, M. Srinivasan, and M. Nicolelis. Real-time prediction of hand trajectory by ensembles of cortical neurons in primates. *Nature*, 408:361–365, 2000.

[21] W. Wu, Y. Gao, E. Bienenstock, J. P. Donoghue, and M. J Black. Bayesian population decoding of motor cortical activity using a Kalman filter. *Neural Computation*, 18(1):80–118, 2006.

[22] W. Wu, A. Shaikhouni, J. P. Donoghue, and M. J. Black. Closed-loop neural control of cursor motion using a Kalman filter. In *Proc. IEEE Engineering in Medicine and Biology Society*, pages 4126–4129, Sep 2004.
